# AUC optimization and the two-sample problem

**Stéphan Clémençon**
Telecom Paristech (TSI) - LTCI UMR Institut Telecom/CNRS 5141
stephan.clemencon@telecom-paristech.fr

**Marine Depecker**
Telecom Paristech (TSI) - LTCI UMR Institut Telecom/CNRS 5141
marine.depecker@telecom-paristech.fr

**Nicolas Vayatis**
ENS Cachan & UniverSud - CMLA UMR CNRS 8536
nicolas.vayatis@cmla.ens-cachan.fr

## Abstract

The purpose of the paper is to explore the connection between multivariate homogeneity tests and AUC optimization. The latter problem has recently received much attention in the statistical learning literature. From the elementary observation that, in the two-sample problem setup, the null assumption corresponds to the situation where the area under the optimal ROC curve is equal to $1/2$, we propose a two-stage testing method based on data splitting. A nearly optimal scoring function in the AUC sense is first learnt from one of the two half-samples. Data from the remaining half-sample are then projected onto the real line and eventually ranked according to the scoring function computed at the first stage. The last step amounts to performing a standard Mann-Whitney Wilcoxon test in the one-dimensional framework. We show that the learning step of the procedure does not affect the consistency of the test as well as its properties in terms of power, provided the ranking produced is accurate enough in the AUC sense. The results of a numerical experiment are eventually displayed in order to show the efficiency of the method.

## 1 Introduction

The statistical problem of testing homogeneity of two samples arises in a wide variety of applications, ranging from bioinformatics to psychometrics through database attribute matching for instance. Practitioners may rely upon a wide range of nonparametric tests for detecting differences in distribution (or location) between two one-dimensional samples, among which tests based on *linear rank statistics*, such as the celebrated Mann-Whitney Wilcoxon test. Being a (locally) optimal procedure, the latter is the most widely used in homogeneity testing. Such rank statistics were originally introduced because they are *distribution-free* under the null hypothesis, thus permitting to set critical values in a non asymptotic fashion for any given level. Beyond this simple fact, the crucial advantage of rank-based tests relies in their asymptotic efficiency in a variety of nonparametric situations. We refer for instance to [15] for an account of asymptotically (locally) uniformly most powerful tests and a comprehensive treatment of asymptotic optimality of $R$-statistics.

In a different context, consider data sampled from a feature space $\mathcal{X} \subset \mathbb{R}^d$ of high dimension with binary label information in $\{-1, +1\}$. The problem of ranking such data, also known as the *bipartite ranking* problem, has recently gained an increasing attention in the machine-learning literature, see

[5, 10, 19]. Here, the goal is to learn, based on a pooled set of labeled examples, how to rank novel data with unknown labels, by means of a *scoring function* $s : \mathcal{X} \to \mathbb{R}$, in order that positive ones appear on top of the list. Over the last few years, this global learning problem has been the subject of intensive research, involving issues related to the design of appropriate criteria reflecting ranking performance or valid extensions of the Empirical Risk Minimization approach (ERM) to this framework [2, 6, 11]. In most applications, the gold standard for measuring the capacity of a scoring function $s$ to discriminate between the class populations however remains the area under the ROC curve criterion (AUC) and most ranking/scoring methods boil down to maximizing its empirical counterpart. The empirical AUC may be viewed as the Mann-Whitney statistic based on the images of the multivariate samples by $s$, see [13, 9, 12, 18].

The purpose of this paper is to investigate how ranking methods for multivariate data with binary labels may be exploited in order to extend the rank-based test approach for testing homogeneity between two samples to a multidimensional setting. Precisely, the testing principle promoted in this paper is described through an extension of the Mann-Whitney Wilcoxon test, based on a preliminary ranking of the data through empirical AUC maximization. The consistency of the test is proved to hold, as soon as the learning procedure is consistent in the AUC sense and its capacity to detect "small" deviations from the homogeneity assumption is illustrated by a simulation example.

The rest of the paper is organized as follows. In Section 2, the homogeneity testing problem is formulated and standard approaches are recalled, with focus on the one-dimensional case. Section 3 highlights the connection of the two-sample problem with optimal ROC curves and gives some insight to our appproach. In Section 4, we describe the testing procedure proposed and set preliminary grounds for its theoretical validity. Simulation results are presented in Section 5 and technical details are deferred to the Appendix.

## 2 The two-sample problem

We start off by setting out the notations needed throughout the paper and formulate the two-sample problem precisely. We recall standard approaches to homogeneity testing. In particular, special attention is paid to the one-dimensional case, for which two-sample linear rank statistics allow for constructing locally optimal tests in a variety of situations.

**Probabilistic setup.** The problem considered in this paper is to test the hypothesis that two independent i.i.d. random samples, valued in $\mathbb{R}^d$ with $d \geq 1$, $X_1^+$, ..., $X_n^+$ and $X_1^-$, ..., $X_m^-$ are identical in distributions. We denote by $G(dx)$ the distribution function of the $X_i^+$'s, while the one of the $X_j^-$'s is denoted by $H(dx)$. We also denote by $\mathbb{P}_{(G,H)}$ the probability distribution on the underlying space. The testing problem is tackled here from a nonparametric perspective, meaning that the distributions $G(dx)$ and $H(dx)$ are assumed to be unknown. We suppose in addition that $G(dx)$ and $H(dx)$ are continuous distributions and the asymptotics are described as follows: we set $N = m + n$ and suppose that $n/N \to p \in (0,1)$ as $n$, $m$ tend to infinity. Formally, the problem is to test the null hypothesis $\mathcal{H}_0 : G = H$ against the alternative $\mathcal{H}_1 : G \neq H$, based on the two data sets. In this paper, we place ourselves in the difficult case where $G$ and $H$ have same support, $\mathcal{X} \subset \mathbb{R}^d$ say.

**Measuring dissimilarity.** A possible approach is to consider a probability (pseudo-)metric $\mathcal{D}$ on the space of probability distributions on $\mathbb{R}^d$. Based on the simple observation that $\mathcal{D}(G, H) = 0$ under the null hypothesis, possible testing procedures consist of computing estimates $\widehat{G}_n$ and $\widehat{H}_m$ of the underlying distributions and rejecting $\mathcal{H}_0$ for "large" values of the statistic $\mathcal{D}(\widehat{G}_n, \widehat{H}_m)$, see [3] for instance. Beyond computational difficulties and the necessity of identifying a proper standardization in order to make the statistic asymptotically pivotal (*i.e.* its limit distribution is parameter free), the major issue one faces when trying to implement such *plug-in* procedures is related to the curse of dimensionality. Indeed, plug-in procedures involve the consistent estimation of distributions on a feature space of possibly very large dimension $d \in \mathbb{N}^*$.

Various metrics or pseudo-metrics can be considered for measuring dissimilarity between two probability distributions. We refer to [17] for an excellent account of metrics in spaces of probability measures and their applications. Typical examples include the chi-square distance, the Kullback-Leibler divergence, the Hellinger distance, the Kolmogorov-Smirnov distance and its generalizations of the

following type

$$\mathrm{MMD}(G, H) = \sup_{f \in \mathcal{F}} \left| \int_{x \in \mathcal{X}} f(x)G(dx) - \int f(x)H(dx) \right|, \qquad (1)$$

where $\mathcal{F}$ denotes a supposedly rich enough class of functions $f : \mathcal{X} \subset \mathbb{R}^d \rightarrow \mathbb{R}$, so that $\mathrm{MMD}(G, H) = 0$ if and only if $G = H$. The quantity (1) is called the *Maximum Mean Discrepancy* in [1], where a unit ball of a reproducing kernel Hilbert space $\mathcal{H}$ is chosen for $\mathcal{F}$ in order to allow for efficient computation of the supremum (1), see also [23]. The view promoted in the present paper for the two-sample problem is very different in nature and is inspired from traditional procedures in the particular one-dimensional case.

**The one-dimensional case.** A classical approach to the two-sample problem in the one-dimensional setup lies in ordering the observed data using the natural order on the real line $\mathbb{R}$ and then basing the decision depending on the ranks of the positive instances among the pooled sample:

$$\forall i \in \{1, \ldots, n\}, \quad \mathcal{R}_i = N F_{n,m}(X_i^+),$$

where $F_{n,m}(t) = (n/N)\widehat{G}_n(t) + (m/N)\widehat{H}_m(t)$, and denoting by $\widehat{G}_n(t) = n^{-1} \sum_{i \leq n} \mathbb{I}\{X_i^+ \leq t\}$ and $\widehat{H}_n(t) = m^{-1} \sum_{i \leq n} \mathbb{I}\{X_i^- \leq t\}$ the empirical counterparts of the cumulative distribution functions $G$ and $H$ respectively. This approach is grounded in invariance considerations, practical simplicity and optimality of tests based on $R$-estimates for this problem, depending on the class of alternative hypotheses considered. Assuming the distributions $G$ and $H$ continuous, the idea underlying such tests lies in the simple fact that, under the null hypothesis, the ranks of positive instances are uniformly distributed over $\{1, \ldots, N\}$. A popular choice is to consider the sum of "positive ranks", leading to the well-known *rank-sum Wilcoxon statistic* [22]

$$\widehat{W}_{n,m} = \sum_{i=1}^{n} \mathcal{R}_i,$$

which is distribution-free under $\mathcal{H}_0$, see Section 6.9 in [15] for further details. We also recall that, the validity framework of the rank-sum test classically extends to the case where some observations are tied (*i.e.* when $G$ and/or $H$ may be degenerate at some points), by assigning the mean rank to ties [4]. We shall denote by $\mathcal{W}_{n,m}$ the distribution of the (average rank version of the) Wilcoxon statistic $\widehat{W}_{n,m}$ under the homogeneity hypothesis. Since tables for the distributions $\mathcal{W}_{n,m}$ are available, no asymptotic approximation result is thus needed for building a test of appropriate level. As it will be recalled below, the test based on the $R$-statistic $\widehat{W}_{n,m}$ has appealing optimality properties for certain classes of alternatives. Although $R$-estimates (*i.e.* functions of the $\mathcal{R}_i$'s) form a very rich collection of statistics, but, for lack of space, we restrict our attention to the two-sample Wilcoxon statistic in this paper.

**Heuristics.** We may now give a first insight into the way we shall tackle the problem in the multi-dimensional case. Suppose that we are able to "project" the multivariate sampling data onto the real line through a certain scoring function $s : \mathbb{R}^d \rightarrow \mathbb{R}$ in order to preserve the possible dissimilarity (considered in a certain specific sense, which we shall discuss below) between the two populations, leading then to "large" values of the score $s(x)$ for the positive instances and "small" values for the negative ones with high probability. Now that the dimension of the problem has been brought down to 1, observations can be ranked and one may perform for instance a basic two-sample Wilcoxon test based on the data sets $s(X_1^+), \ldots, s(X_n^+)$ and $s(X_1^-), \ldots, s(X_m^-)$.

**Remark 1** (LEARNING A STUDENT t TEST.) We point out that it is precisely the task Linear Discriminant Analysis (LDA) tries to performs, in a restrictive Gaussian framework however (when $G$ and $H$ are normal distributions with same covariance structure namely). In order to test deviations from the homogeneity hypothesis on the basis of the original samples, one may consider applying a univariate Student t test based on the "projected" data $\{\widehat{\delta}(X_i^+) : 1 \leq i \leq n\}$ and $\{\widehat{\delta}(X_i^-) : 1 \leq i \leq m\}$, where $\widehat{\delta}$ denotes the empirical discriminant function, this may be shown as an appealing alternative to multivariate extensions of the standard t test [14].

The goal of this paper is to show how to exploit recent advances in ROC/AUC optimization for extending this heuristics to more general situations than the parametric one mentioned above.

# 3   Connections with bipartite ranking

ROC curves are among the most widely used graphical tools for visualizing the dissimilarity between two one-dimensional distributions in a large variety of applications such as anomaly detection in signal analysis, medical diagnosis, information retrieval, *etc.* As this concept is at the heart of the ranking issue in the binary setting, which forms the first stage of the testing procedure sketched above, we recall its definition precisely.

**Definition 1** (ROC **curve**) *Let $g$ and $h$ be two cumulative distribution functions on $\mathbb{R}$. The* ROC *curve related to the distributions $g(dt)$ and $h(dt)$ is the graph of the mapping:*

$$\mathrm{ROC}\,((\mathrm{g},\mathrm{h}),\ \cdot):\alpha\in[0,1]\mapsto 1-\mathrm{g}\circ\mathrm{h}^{-1}(1-\alpha),$$

*denoting by $f^{-1}(u)=\inf\{t\in\mathbb{R}:f(t)\geq u\}$ the generalized inverse of any càd-làg function $f:\mathbb{R}\to\mathbb{R}$. When the distributions $g(dt)$ and $h(t)$ are continuous, it can alternatively be defined as the parametric curve $t\in\mathbb{R}\mapsto(1-h(t),1-g(t))$.*

One may show that $\mathrm{ROC}\,((\mathrm{g},\mathrm{h}),\ \cdot)$ is above the diagonal $\Delta:\alpha\in[0,1]\mapsto\alpha$ of the ROC space if and only if the distribution $g$ is stochastically larger than $h$ and it is concave as soon as the likelihood ratio $dg/dh$ is increasing. When $g(dt)$ and $h(dt)$ are both continuous, the curves $\mathrm{ROC}((\mathrm{g},\mathrm{h}),.)$ and $\mathrm{ROC}((\mathrm{h},\mathrm{g}),.)$ are symmetric with respect to the diagonal of the ROC space with slope equal to one. Refer to [9] for a detailed list of properties of ROC curves.

The notion of ROC curve provides a functional measure of dissimilarity between distributions on $\mathbb{R}$: the closer to the corners of the unit square the curve $\mathrm{ROC}\,((\mathrm{g},\mathrm{h}),\ \cdot)$ is, the more dissimilar the distributions $g$ and $h$ are. For instance, it exactly coincides with the upper left-hand corner of the unit square, namely the curve $\alpha\in[0,1]\mapsto\mathbb{I}\{\alpha\in]0,1]\}$, when there exists $l\in\mathbb{R}$ such that the support of distribution $g(dt)$ is a subset of $[l,\ \infty[$, while $]l,-\infty,]$ contains the support of $h$. In contrast, it merges with the diagonal $\Delta$ when $g=h$. Hence, distance of $\mathrm{ROC}\,((\mathrm{g},\mathrm{h}),\ \cdot)$ to the diagonal may be naturally used to quantify departure from the homogeneous situation. The $L_1$-norm provides a convenient way of measuring such a distance, leading to the classical AUC criterion (AUC standing for *area under the* ROC *curve*):

$$\mathrm{AUC}(\mathrm{g},\mathrm{h})=\int_{\alpha=0}^{1}\mathrm{ROC}\,((\mathrm{g},\mathrm{h}),\ \alpha)\,\mathrm{d}\alpha.$$

The popularity of this summary quantity arises from the fact that it can be interpreted in a probabilistic fashion, and may be viewed as a distance between the locations of the two distributions. In this respect, we recall the following result.

**Proposition 1** *Let $g$ and $h$ be two distributions on $\mathbb{R}$. We have:*

$$\mathrm{AUC}(\mathrm{g},\mathrm{h})\quad=\quad\mathbb{P}\left\{Z>Z'\right\}+\frac{1}{2}\mathbb{P}\left\{Z=Z'\right\}=\frac{1}{2}+\mathbb{E}[h(Z)]-\mathbb{E}[g(Z')],$$

*where $Z$ and $Z'$ denote independent random variables, drawn from $g(dt)$ and $h(dt)$ respectively.*

We recall that the homogeneous situation corresponds to the case where $\mathrm{AUC}(\mathrm{g},\mathrm{h})=1/2$ and the Mann-Withney statistic [16]

$$U_{n,m}=\frac{1}{nm}\sum_{i=1}^{n}\sum_{j=1}^{m}\left(\mathbb{I}\{X_j^-<X_i^+\}+\frac{1}{2}\mathbb{I}\{X_j^-=X_i^+\}\right)$$

is exactly the empirical counterpart of $\mathrm{AUC}(\mathrm{g},\mathrm{h})$. It yields exactly the same statistical decisions as the two-sample Wilcoxon statistic, insofar they are related as follows:

$$W_{n,m}=nm\widehat{U}_{n,m}+n(n+1)/2.$$

For this reason, the related test of hypotheses is called Mann-Whitney Wilcoxon test (*MWW*).

**Multidimensional extension.** In the multivariate setup, the notion of ROC curve can be extended the following way. Let $H(dx)$ and $G(dx)$ be two given distributions on $\mathbb{R}^d$ and $\mathcal{S}=\{s:\mathcal{X}\to\mathbb{R}\mid$

$s$ Borel measurable}. For any *scoring function* $s \in \mathcal{S}$, we denote by $H_s(dt)$ and $G_s(t)$ the images of $H(dx)$ and $G(x)$ by the mapping $s(x)$. In addition, we set for all $s \in \mathcal{S}$:

$$\text{ROC}(s,.) = \text{ROC}((G_s, H_s), \ .) \text{ and } \text{AUC}(s) = \text{AUC}(G_s, H_s).$$

Clearly, the families of univariate distributions $\{G_s\}_{s \in \mathcal{S}}$ and $\{H_s\}_{s \in \mathcal{S}}$ entirely characterize the multivariate probability measures $G$ and $H$. One may thus consider evaluating the dissimilarity between $H(dx)$ and $G(dx)$ on $\mathbb{R}^d$ through the family of curves $\{\text{ROC}(s,.)\}_{s \in \mathcal{S}}$ or through the collection of scalar values $\{\text{AUC}(s)\}_{s \in \mathcal{S}}$. Going back to the homogeneity testing problem, the null assumption may be reformulated as

"$\mathcal{H}_0 : \forall s \in \mathcal{S}, \text{AUC}(s) = 1/2$" *versus* "$\mathcal{H}_1 : \exists s \in \mathcal{S}$ such that $\text{AUC}(s) > 1/2$".

The next result, following from standard Neyman-Pearson type arguments, shows that the supremum $\sup_{s \in \mathcal{S}} \text{AUC}(s)$ is attained by increasing transforms of the likelihood ratio $\phi(x) = dG/dH(x)$, $x \in \mathcal{X}$. Scoring functions with largest AUC are natural candidates for detecting the alternative $\mathcal{H}_1$.

**Theorem 1** (OPTIMAL ROC CURVE.) *The set of $\mathcal{S}^* = \{T \circ \phi \mid T : \mathbb{R} \to \mathbb{R}$ strictly increasing $\}$ defines the collection of optimal scoring functions in the sense that: $\forall s \in \mathcal{S}$,*

$$\forall \alpha \in [0,1], \ \ \text{ROC}(s, \alpha) \leq \text{ROC}^*(\alpha) \text{ and } \text{AUC}(s) \leq \text{AUC}^*,$$

*with the notations $\text{ROC}^*(.) = \text{ROC}(s^*,.)$ and $\text{AUC}^* = \text{AUC}(s^*)$ for $s^* \in \mathcal{S}^*$.*

Refer to Proposition 4's proof in [9] for a detailed argument. Notice that, as $dG/dH(X) = dG_\phi(X)/dH_\phi(\phi(X))$, replacing $X$ by $s^*(X)$ with $s^* \in \mathcal{S}^*$ leaves the optimal ROC curve untouched. The following corollary is straightforward.

**Corollary 1** *For any $s \in \mathcal{S}^*$, we have:* $\sup_{s \in \mathcal{S}} |\text{AUC}(s) - 1/2| = \text{AUC}(s^*) - 1/2.$

Consequently, the homogeneity testing problem may be seen as closely related to the problem of estimating the optimal $\text{AUC}^*$, since it may be re-formulated as follows:

"$\mathcal{H}_0 : \text{AUC}^* = 1/2$" *versus* "$\mathcal{H}_1 : \text{AUC}^* > 1/2$".

Knowing how a single optimal scoring function $s^* \in \mathcal{S}^*$ ranks observations drawn from a mixture of $G$ and $H$ is sufficient for detecting departure from the homogeneity hypothesis in an optimal fashion, the MWW statistic computed from the $(s^*(X_i^+), s^*(X_j^-))$'s being an asymptotically efficient estimate of $\text{AUC}^*$ and thus yields an asymptotically (locally) uniformly most powerful test.

Let $F(dx) = pG(dx) + (1-p)H(dx)$ and denote by $F_s(dt)$ the image of the distribution $F$ by $s \in \mathcal{S}$. Notice that, for any $s^* \in \mathcal{S}^*$, the scoring function $S^* = F_{s^*} \circ s^*$ is still optimal and the score variable $S^*(X)$ is uniformly distributed on $[0,1]$ under the mixture distribution $F$ (in addition, it may be easily shown to be independent from $s^* \in \mathcal{S}^*$). Observe in addition that $\text{AUC}^* - 1/2$ may be viewed as the Earth Mover's distance between the class distributions $H_{S^*}$ and $G_{S^*}$ for this "normalization":

$$\text{AUC}^* - 1/2 = \int_{t=0}^{1} \{H_{S^*}(t) - G_{S^*}(t)\} \, dt.$$

**Empirical** AUC **maximization.** A natural way of inferring the value of $\text{AUC}^*$ and/or selecting a scoring function $\hat{s}$ with AUC nearly as large as $\text{AUC}^*$ is to maximize an empirical version of the AUC criterion over a set $\mathcal{S}_0$ of scoring function candidates. We assume that the class $\mathcal{S}_0$ is sufficiently rich in order to guarantee that the bias $\text{AUC}^* - \sup_{s \in \mathcal{S}_0} \text{AUC}(s)$ is small, and its complexity is controlled (when measured for instance by the VC dimension of the collection of sets $\{\{x \in \mathcal{X} : s(x) \geq t\}, (s,t) \in \mathcal{S}_0 \times \mathbb{R}\}$ as in [7] or by the order of magnitude of conditional Rademacher averages as in [6]). We recall that, under such assumptions, universal consistency results have been established for empirical AUC maximizers, together with distribution-free generalization bounds, see [2, 6] for instance. We point out that this approach can be extended to other relevant ranking criteria. The contours of a theory guaranteeing the statistical performance of the ERM approach for empirical risk functionals defined by $R$-estimates have been sketched in [8].

## 4 The two-stage testing procedure

Assume that data have been split into two subsamples: the first data set $\mathcal{D}_{n_0, m_0} = \{X_1^+, \ldots, X_{n_0}^+\} \cup \{X_1^-, \ldots, X_{m_0}^-\}$ will be used for deriving a scoring function on $\mathcal{X}$ and the second data set $\mathcal{D}'_{n_1, m_1} = \{X_{n_0+1}^+, \ldots, X_{n_0+n_1}^+\} \cup \{X_{m_0+1}^-, \ldots, X_{m_0+m_1}^-\}$ will serve to compute a pseudo- two-sample Wilcoxon test statistic from the ranked data. We set $N_0 = n_0 + m_0$ and $N_1 = n_1 + m_1$ and suppose that $n_i/N_i \to p$ as $n_i$ and $m_i$ tend to infinity for $i \in \{0, 1\}$.

Let $\alpha \in (0, 1)$. The testing procedure at level $\alpha$ is then performed in two steps, as follows.

---

SCORE-BASED RANK-SUM WILCOXON TEST

1. **Ranking.** From dataset $\mathcal{D}_{n_0, m_0}$, perform empirical AUC maximization over $\mathcal{S}_0 \subset \mathcal{S}$, yielding the scoring function $\hat{s}(x) = \hat{s}_{n_0, m_0}(x)$. Compute the ranks of data with positive labels among the sample $\mathcal{D}'_{n_1, m_1}$, once sorted by increasing order of magnitude of their score:

$$\widehat{\mathcal{R}}_i = N_1 \hat{S}(X_{n_0+i}^+) \text{ for } 1 \le i \le n_1,$$

where $\widehat{F}_{\hat{s}}(t) = N_1^{-1} \left( \sum_{i=1}^{n_1} \mathbb{I}\{\hat{s}(X_{n_0+i}^+) \le t\} + \sum_{j=1}^{m_1} \mathbb{I}\{\hat{s}(X_{m_0+j}^-) \le t\} \right)$ and $\hat{S} = \widehat{F}_{\hat{s}} \circ \hat{s}$.

2. **Rank-sum Wilcoxon test.** Reject the homogeneity hypothesis $\mathcal{H}_0$ when:

$$\widehat{W}_{n_1, m_1} \ge Q_{n_1, m_1}(\alpha),$$

where $\widehat{W}_{n_1, m_1} = \sum_{i=1}^{n_1} \widehat{\mathcal{R}}_i$ and $Q_{n_1, m_1}(\alpha)$ denotes the $(1-\alpha)$-quantile of distribution $\mathcal{W}_{n_1, m_1}$.

---

The next result shows that the learning step does not affect the consistency property, provided it outputs a universally consistent scoring rule.

**Theorem 2** *Let $\alpha \in (0, 1/2)$ and suppose that the ranking/scoring method involved at step 1 yields a universally consistent scoring rule $\hat{s}$ in the AUC sense. The score-based rank-sum Wilcoxon test*

$$\Phi = \mathbb{I}\left\{ \widehat{W}_{n_1, m_1} \ge Q_{n_1, m_1}(\alpha) \right\}$$

*is universally consistent as $n_i$ and $m_i$ tend to $\infty$ for $i \in \{0, 1\}$ at level $\alpha$, in the following sense.*

1. *It is of level $\alpha$ for all $n_i$ and $m_i$, $i \in \{0, 1\}$: $\mathbb{P}_{(H, H)}\{\Phi = +1\} \le \alpha$ for any $H(dx)$.*

2. *Its power converges to 1 as $n_i$ and $m_i$, $i \in \{0, 1\}$, tend to infinity for every alternative: $\lim_{n_i, m_i \to \infty} \mathbb{P}_{(G, H)}\{\Phi = +1\} = 1$ for every pair of distinct distributions $(G, H)$.*

**Remark 2** (CONVERGENCE RATES.) Under adequate complexity assumptions on the set $\mathcal{S}_0$ over which empirical AUC maximization or one of its variants is performed, distribution-free rate bounds for the generalization ability of scoring rules may be established in terms of AUC, see Corollary 6 in [2] or Corollary 3 in [6]. As shown by a careful examination of Theorem 2, this permits to derive a convergence rate for the decay of the score-based type II error of MWW under any given alternative $(G, H)$, when combined with the Berry-Esseen theorem for two-sample $U$-statistics. For instance, if a typical $1/\sqrt{N_0}$ rate bound holds for $\hat{s}(x)$, one may show that choosing $N_1 \sim N_0$ then yields a rate of order $O_{\mathbb{P}_{(G, H)}}(1/\sqrt{N_0})$.

**Remark 3** (INFINITE-DIMENSIONAL FEATURE SPACE.) We point out that the method presented here is by no means restricted to the case where $\mathcal{X}$ is of finite dimension, but may be applied to functional input data, provided an AUC-consistent ranking procedure can be applied in this context.

## 5 Numerical examples

The procedure proposed above is extremely simple once the delicate AUC maximization stage is performed. A stunning property is the fact that critical thresholds are set automatically, with no reference to the data. We firts consider a low-dimensional toy experiment and display some numerical results. Two independent i.i.d. samples of equal size $m = n = N/2$ have been generated from two conditional 4-dimensional gaussian distributions on the hypercube $[-2, 2]^4$. Their parameters

are denoted by $\mu_+$ and $\mu_-$ for the means and $\Gamma$ is their common covariance matrix. Three cases have been considered. The first example corresponds to a homogeneous situation: $\mu_+ = \mu_- = \mu_1$ where $\mu_1 = (-0.96, -0.83, 0.29, -1.34)$ and the upper diagonals of $\Gamma_1$ are $(6.52, 3.84, 4.72, 3.1)$, $(-1.89, 3.56, 1.52)$, $(-3.2, 0.2)$ and $(-2.6)$. In the second example, we test homogeneity under an alternative, "fairly far" from $\mathcal{H}_0$, where $\mu_- = \mu_1$, $\mu_+ = (0.17, -0.24, 0.04, -1.02)$ and $\Gamma$ as before. Eventually, the third example corresponds to a much more difficult problem, "close" to $\mathcal{H}_0$, where $\mu_- = (1.19, -1.20, -0.02, -0.16)$, $\mu_+ = (1.08, -1.18, -0.1, -0.06)$ and the upper diagonals of $\Gamma$ are $(1.83, 6.02, 0.69, 4.99)$, $(-0.65, -0.31, 1.03)$, $(-0.54, -0.03)$ and $(-1.24)$. The difficulty of each of these examples is illustrated by Fig. 2 in terms of (optimal) ROC curve. The table in Fig. 2 gives Monte-Carlo estimates of the power of three testing procedures when $\alpha = 0.05$ (averaged over $B = 150$ replications): 1) the score-based MWW test, where ranking is performed using the scoring function output by a run of the TREERANK algorithm [9] on a training sample $\mathcal{D}_{n_0, m_0}$, 2) the LDA-based Student test sketched in Remark 1 and 3) a bootstrap version of the MMD-test with a Gaussian RBF Kernel proposed in [1].

| DataSet | Sample size $(m_0, m_1)$ | LDA-Student | Score-based MWW | MMD |
|---------|--------------------------|-------------|-----------------|-----|
| *Ex. 1* | (500,500) | 6% | 1% | 5% |
| *Ex. 2* | (500,500) | 99% | 99% | 99% |
|         | (2000,1000) | 75% | 45% | 30% |
| *Ex. 3* | (3000,2000) | 98% | 73% | 65% |

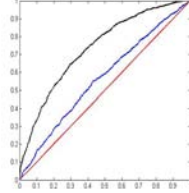

Figure 1: Powers and ROC curves describing the "distance" to $\mathcal{H}_0$ for each situation: example 1 (red), example 2 (black) and example 3 (blue).

In the second series of experimental results, gaussian distributions with same covariance matrix on $\mathbb{R}^d$ are generated, with larger values for the input space dimension $d \in \{10, 30\}$. We considered several problems at given toughness. The increasing difficulty of the testing problems considered is controlled through the euclidian distance between the means $\Delta\mu = ||\mu_+ - \mu_-||$ and is described by Fig. 2, which depicts the related ROC curves, corresponding to situations where $\Delta\mu \in \{0.2, 0.1, 0.08, 0.05\}$. On these examples, we compared the performance of four methods at level $\alpha = 0.05$: the score-based MWW test, where ranking is again performed using the scoring function output by a run of the TREERANK algorithm on a training sample $\mathcal{D}_{n_0, m_0}$, the KFDA test proposed in [23], a bootstrap version of the MMD-test with a Gaussian RBF Kernel ($MMD$) and another version, with moment matching to Pearson curves ($MMD_{mom}$), using also with a Gaussian RBF kernel (see [1]). Monte-Carlo estimates of the corresponding powers are given in the Table displayed in Fig. 2.

## 6 Conclusion

We have provided a sound strategy, involving a preliminary bipartite ranking stage, to extend classical approaches for testing homogeneity based on ranks to a multidimensional setup. Consistency of the extended version of the popular MWW test has been established, under the assumption of universal consistency of the ranking method in the AUC sense. This principle can be applied to other $R$-statistics, standing as natural criteria for the bipartite ranking problem [8]. Beyond the illustrative preliminary simulation example displayed in this paper, we intend to investigate the relative efficiency of such tests with respect to other tests standing as natural candidates in this setup.

## Appendix - Proof of Theorem 2

Observe that, conditioned upon the first sample $\mathcal{D}_{n_0, m_0}$, the statistic $\widehat{W}_{n_1, m_1}$ is distributed according to $\mathcal{W}_{n_1, m_1}$ under the null hypothesis. For any distribution $H$, we thus have: $\forall \alpha \in (0, 1/2)$,

$$\mathbb{P}_{(H,H)} \left\{ \widehat{W}_{n_1, m_1} > Q_{n_1, m_1}(\alpha) \mid \mathcal{D}_{n_0, m_0} \right\} \leq \alpha.$$

Taking the expectation, we obtain that the test is of level $\alpha$ for all $n$, $m$.

| Dim. d | $MMD_{boot}$ | $MMD_{mom}$ | Kfda | Sc.based MWW |
|--------|--------------|-------------|------|--------------|
| case 1 : $\Delta\mu = 0.2$ | | | | |
| $d = 10$ | 86% | 86% | 64% | 90% |
| $d = 30$ | 54% | 58% | 36% | 85% |
| case 1 : $\Delta\mu = 0.1$ | | | | |
| $d = 10$ | 20% | 20% | 20% | 58% |
| $d = 30$ | 9% | 7% | 15% | 47% |
| case 3 : $\Delta\mu = 0.08$ | | | | |
| $d = 10$ | 19% | 19% | 16% | 42% |
| $d = 30$ | 5% | 7% | 9% | 32% |
| case 4 : $\Delta\mu = 0.05$ | | | | |
| $d = 10$ | 11% | 13% | 13% | 18% |
| $d = 30$ | 6% | 6% | 8% | 16% |

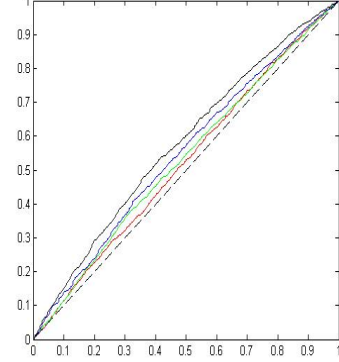

Figure 2: Power estimates and ROC curves describing the "distance" to $\mathcal{H}_0$ for each situation: case 1 (black), case 2 (blue), case 3 (green) and case 4 (red).

For any $s \in \mathcal{S}$, denote by $U_{n_1,m_1}(s)$ the empirical AUC of $s$ evaluated on the sample $\mathcal{D}'_{n_1,m_1}$. Recall first that it follows from the two-sample $U$-statistic theorem (see [20]) that:

$$\sqrt{N}\{U_{n_1,m_1}(s) - \mathrm{AUC(s)}\} = \frac{\sqrt{N_1}}{n_1} \sum_{i=1}^{n_1} \left\{ \mathrm{H_s}(s(\mathrm{X}^+_{i+n_0})) - \mathbb{E}[\mathrm{H_s}(s(\mathrm{X}^+_1))] \right\}$$

$$- \frac{\sqrt{N_1}}{m_1} \sum_{j=1}^{m_1} \left\{ G_s(s(X^-_{j+m_0})) - \mathbb{E}[G_s(s(X^-_1))] \right\} + o_{\mathbb{P}_{(G,H)}}(1),$$

as $n$, $m$ tend to infinity. In particular, for any pair of distributions $(G, H)$, the centered random variable $\sqrt{N}\{U_{n_1,m_1}(s) - \mathrm{AUC(s)}\}$ is asymptotically normal with limit variance $\sigma_s^2(G, H) = \mathrm{Var}(H_s(s(X^+_1)))/p + \mathrm{Var}(G_s(s(X^-_1)))/(1-p)$ under $\mathbb{P}_{(G,H)}$. Notice that $\sigma_s^2(H, H) = 1/(12p(1-p))$ for any $s \in \mathcal{S}$ such that the distribution $H_s(dt)$ is continuous. Refer to Theorem 12.4 in [21] for further details.

We now place ourselves under an alternative hypothesis described by a pair of distinct distribution $(G, H)$, so that $\mathrm{AUC}^* > 1/2$. Setting $\widehat{U}_{n_1,m_1} = U_{n_1,m_1}(\hat{s})$ and decomposing $\mathrm{AUC}^* - \widehat{U}_{n_1,m_1}$ as the sum of the *deficit of* AUC of $\hat{s}(x)$, $\mathrm{AUC}^* - \mathrm{AUC}(\hat{s})$ namely, and the deviation $\mathrm{AUC}(\hat{s}) - \widehat{U}_{n_1,m_1}$ evaluated on the sample $\mathcal{D}'_{n_1,m_1}$, type II error of $\Phi$ given by $\mathbb{P}_{(G,H)}\left\{\widehat{W}_{n_1,m_1} \leq Q_{n_1,m_1}(\alpha)\right\}$ may be bounded by:

$$\mathbb{P}_{(G,H)}\left\{\sqrt{N_1}\left(\widehat{U}_{n_1,m_1} - \mathrm{AUC}(\hat{s})\right) \leq \epsilon_{n_1,m_1}(\alpha)\right\}$$

$$+ \mathbb{P}_{(G,H)}\left\{\sqrt{N_1}\left(\mathrm{AUC}(\hat{s}) - \mathrm{AUC}^*\right) \leq \epsilon_{n_1,m_1}(\alpha)\right\},$$

where

$$\epsilon_{n_1,m_1}(\alpha) = \sqrt{N_1}\left(\frac{Q_{n_1,m_1}(\alpha)}{n_1 m_1} - \frac{n_1 + 1}{2m_1} - \frac{1}{2}\right) - \sqrt{N_1}(\mathrm{AUC}^* - \frac{1}{2}).$$

Observe that, by virtue of the CLT recalled above, $\sqrt{N_1}(Q_{n_1,m_1}(\alpha)/(n_1 m_1) - (n_1 + 1)/(2m_1))$ converges to $z_\alpha/\sqrt{12p(1-p)}$. Now, the fact that type II error of $\Phi$ converges to zero as $n_i$ and $m_i$ tend to $\infty$ for $i \in \{0, 1\}$ immediately follows from the assumption in regards to the AUC of $\hat{s}(x)$ universal consistency and the CLT for two-sample $U$-statistics combined with the theorem of dominated convergence. Due to space limitations, details are omitted.

# References

[1] M.J. Rasch B. Scholkopf A. Smola A. Gretton, K.M. Borgwardt. A kernel method for the two-sample problem. In *Advances in Neural Information Processing Systems 19*. MIT Press, Cambridge, MA, 2007.

[2] S. Agarwal, T. Graepel, R. Herbrich, S. Har-Peled, and D. Roth. Generalization bounds for the area under the ROC curve. *J. Mach. Learn. Res.*, 6:393–425, 2005.

[3] G. Biau and L. Gyorfi. On the asymptotic properties of a nonparametric $l_1$-test statistic of homogeneity. *IEEE Transactions on Information Theory*, 51(11):3965–3973, 2005.

[4] Y.K. Cheung and J.H. Klotz. The Mann Whitney Wilcoxon distribution using linked list. *Statistica Sinica*, 7:805–813, 1997.

[5] S. Clémençon, G. Lugosi, and N. Vayatis. Ranking and scoring using empirical risk minimization. In P. Auer and R. Meir, editors, *Proceedings of COLT 2005*, volume 3559 of *Lecture Notes in Computer Science*, pages 1–15. Springer, 2005.

[6] S. Clémençon, G. Lugosi, and N. Vayatis. Ranking and empirical risk minimization of U-statistics. *The Annals of Statistics*, 36(2):844–874, 2008.

[7] S. Clémençon and N. Vayatis. Ranking the best instances. *Journal of Machine Learning Research*, 8:2671–2699, 2007.

[8] S. Clémençon and N. Vayatis. Empirical performance maximization based on linear rank statistics. In *Advances in Neural Information Processing Systems*, volume 3559 of *Lecture Notes in Computer Science*, pages 1–15. Springer, 2009.

[9] S. Clémençon and N. Vayatis. Tree-based ranking methods. *IEEE Transactions on Information Theory*, 55(9):4316–4336, 2009.

[10] W.W. Cohen, R.E. Schapire, and Y. Singer. Learning to order things. In *NIPS '97: Proceedings of the 1997 conference on Advances in neural information processing systems 10*, pages 451–457, Cambridge, MA, USA, 1998. MIT Press.

[11] C. Cortes and M. Mohri. AUC optimization vs. error rate minimization. In S. Thrun, L. Saul, and B. Schölkopf, editors, *Advances in Neural Information Processing Systems 16*. MIT Press, Cambridge, MA, 2004.

[12] C. Ferri, P.A. Flach, and J. Hernández-Orallo. Learning decision trees using the area under the roc curve. In *ICML '02: Proceedings of the Nineteenth International Conference on Machine Learning*, pages 139–146, 2002.

[13] Y. Freund, R. D. Iyer, R. E. Schapire, and Y. Singer. An efficient boosting algorithm for combining preferences. *Journal of Machine Learning Research*, 4:933–969, 2003.

[14] S. Kotz and S. Nadarajah. *Multivariate t Distributions and Their Applications*. Cambridge University Press, 2004.

[15] E.L. Lehmann and J. P. Romano. *Testing Statistical Hypotheses*. Springer, 2005.

[16] H.B. Mann and D.R. Whitney. On a test of whether one of two random variables is stochastically larger than the other. *Ann. Math. Stat.*, 18:50–60, 1947.

[17] A. Rachev. *Probability Metrics and the Stability of Stochastic Models*. Wiley, 1991.

[18] A. Rakotomamonjy. Optimizing Area Under Roc Curve with SVMs. In *Proceedings of the First Workshop on ROC Analysis in AI*, 2004.

[19] C. Rudin, C. Cortes, M. Mohri, and R. E. Schapire. Margin-based ranking and boosting meet in the middle. In P. Auer and R. Meir, editors, *Proceedings of COLT 2005*, volume 3559 of *Lecture Notes in Computer Science*, pages 63–78. Springer, 2005.

[20] R.J. Serfling. *Approximation theorems of mathematical statistics*. Wiley, 1980.

[21] A.K. van der Vaart. *Asymptotic Analysis*. Cambridge University Press, 1998.

[22] F. Wilcoxon. Individual comparisons by ranking methods. *Biometrics*, 1:80–83, 1945.

[23] E. Moulines Z. Harchaoui, F. Bach. Testing for homogeneity with kernel Fischer discriminant analysis. In *Advances in Neural Information Processing Systems 20*. MIT Press, Cambridge, MA, 2008.

